# Maximum Likelihood Estimation of a Stochastic Integrate-and-Fire Neural Model[*]

**Jonathan W. Pillow, Liam Paninski, and Eero P. Simoncelli**
Howard Hughes Medical Institute
Center for Neural Science
New York University
*{pillow, liam, eero}@cns.nyu.edu*

## Abstract

Recent work has examined the estimation of models of stimulus-driven neural activity in which some linear filtering process is followed by a nonlinear, probabilistic spiking stage. We analyze the estimation of one such model for which this nonlinear step is implemented by a noisy, leaky, integrate-and-fire mechanism with a spike-dependent after-current. This model is a biophysically plausible alternative to models with Poisson (memory-less) spiking, and has been shown to effectively reproduce various spiking statistics of neurons in vivo. However, the problem of estimating the model from extracellular spike train data has not been examined in depth. We formulate the problem in terms of maximum likelihood estimation, and show that the computational problem of maximizing the likelihood is tractable. Our main contribution is an algorithm and a proof that this algorithm is guaranteed to find the global optimum with reasonable speed. We demonstrate the effectiveness of our estimator with numerical simulations.

A central issue in computational neuroscience is the characterization of the functional relationship between sensory stimuli and neural spike trains. A common model for this relationship consists of linear filtering of the stimulus, followed by a nonlinear, probabilistic spike generation process. The linear filter is typically interpreted as the neuron's "receptive field," while the spiking mechanism accounts for simple nonlinearities like rectification and response saturation. Given a set of stimuli and (extracellularly) recorded spike times, the characterization problem consists of estimating both the linear filter and the parameters governing the spiking mechanism.

One widely used model of this type is the Linear-Nonlinear-Poisson (LNP) cascade model, in which spikes are generated according to an inhomogeneous Poisson process, with rate determined by an instantaneous ("memoryless") nonlinear function of the filtered input. This model has a number of desirable features, including conceptual simplicity and computational tractability. Additionally, reverse correlation analysis provides a simple unbiased estimator for the linear filter [5], and the properties of estimators (for both the linear filter and static nonlinearity) have been thoroughly analyzed, even for the case of highly non-symmetric or "naturalistic" stimuli [12]. One important drawback of the LNP model,

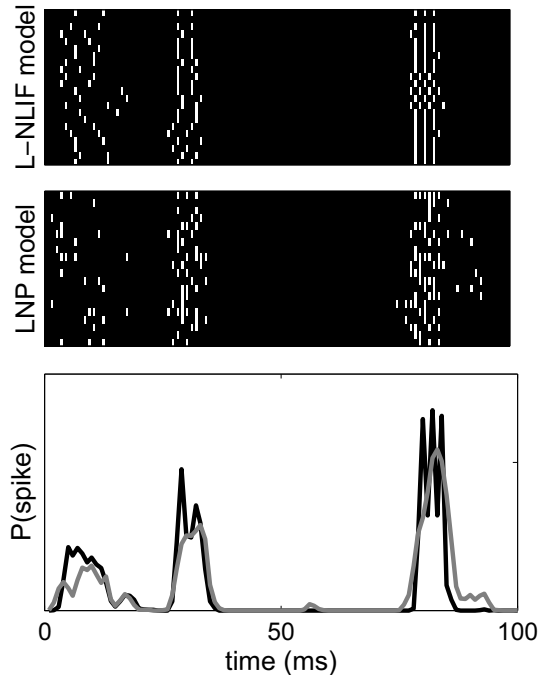

Figure 1: Simulated responses of L-NLIF and LNP models to 20 repetitions of a fixed 100-ms stimulus segment of temporal white noise. **Top:** Raster of responses of L-NLIF model, where $\sigma_{noise}/\sigma_{signal} = 0.5$ and $g$ gives a membrane time constant of 15 ms. The top row shows the fixed (deterministic) response of the model with $\sigma_{noise}$ set to zero. **Middle:** Raster of responses of LNP model, with parameters fit with standard methods from a long run of the L-NLIF model responses to non-repeating stimuli. **Bottom:** (Black line) Post-stimulus time histogram (PSTH) of the simulated L-NLIF response. (Gray line) PSTH of the LNP model. Note that the LNP model fails to preserve the fine temporal structure of the spike trains, relative to the L-NLIF model.

however, is that Poisson processes do not accurately capture the statistics of neural spike trains [2, 9, 16, 1]. In particular, the probability of observing a spike is not a functional of the stimulus only; it is also strongly affected by the recent history of spiking.

The leaky integrate-and-fire (LIF) model provides a biophysically more realistic spike mechanism with a simple form of spike-history dependence. This model is simple, well-understood, and has dynamics that are entirely linear except for a nonlinear "reset" of the membrane potential following a spike. Although this model's overriding linearity is often emphasized (due to the approximately linear relationship between input current and firing rate, and lack of active conductances), the nonlinear reset has significant functional importance for the model's response properties. In previous work, we have shown that standard reverse correlation analysis fails when applied to a neuron with deterministic (noise-free) LIF spike generation; we developed a new estimator for this model, and demonstrated that a change in leakiness of such a mechanism might underlie nonlinear effects of contrast adaptation in macaque retinal ganglion cells [15]. We and others have explored other "adaptive" properties of the LIF model [17, 13, 19].

In this paper, we consider a model consisting of a linear filter followed by noisy LIF spike generation with a spike-dependent after-current; this is essentially the standard LIF model driven by a noisy, filtered version of the stimulus, with an additional current waveform injected following each spike. We will refer to this as the the "L-NLIF" model. The probabilistic nature of this model provides several important advantages over the deterministic version we have considered previously. First, an explicit noise model allows us to couch the problem in the terms of classical estimation theory. This, in turn, provides a natural "cost function" (likelihood) for model assessment and leads to more efficient estimation of the model parameters. Second, noise allows us to explicitly model neural firing statistics, and could provide a rigorous basis for a metric distance between spike trains, useful in other contexts [18]. Finally, noise influences the behavior of the model itself, giving rise to

phenomena not observed in the purely deterministic model [11].

Our main contribution here is to show that the maximum likelihood estimator (MLE) for the L-NLIF model is computationally tractable. Specifically, we describe an algorithm for computing the likelihood function, and prove that this likelihood function contains no non-global maxima, implying that the MLE can be computed efficiently using standard ascent techniques. The desirable statistical properties of this estimator (e.g. consistency, efficiency) are all inherited "for free" from classical estimation theory. Thus, we have a compact and powerful model for the neural code, and a well-motivated, efficient way to estimate the parameters of this model from extracellular data.

## The Model

We consider a model for which the (dimensionless) subthreshold voltage variable $V$ evolves according to

$$dV = \left( -gV(t) + \vec{k} \cdot \vec{x}(t) + \sum_{j=0}^{i-1} h(t - t_j) \right) dt + \sigma N_t, \tag{1}$$

and resets to $V_r$ whenever $V = 1$. Here, $g$ denotes the leak conductance, $\vec{k} \cdot \vec{x}(t)$ the projection of the input signal $\vec{x}(t)$ onto the linear kernel $\vec{k}$, $h$ is an "afterpotential," a current waveform of fixed amplitude and shape whose value depends only on the time since the last spike $t_{i-1}$, and $N_t$ is an unobserved (hidden) noise process with scale parameter $\sigma$. Without loss of generality, the "leak" and "threshold" potential are set at 0 and 1, respectively, so the cell spikes whenever $V = 1$, and $V$ decays back to 0 with time constant $1/g$ in the absence of input. Note that the nonlinear behavior of the model is completely determined by only a few parameters, namely $\{g, \sigma, V_r\}$, and $h$ (where the function $h$ is allowed to take values in some low-dimensional vector space). The dynamical properties of this type of "spike response model" have been extensively studied [7]; for example, it is known that this class of models can effectively capture much of the behavior of apparently more biophysically realistic models (e.g. Hodgkin-Huxley).

Figures 1 and 2 show several simple comparisons of the L-NLIF and LNP models. In 1, note the fine structure of spike timing in the responses of the L-NLIF model, which is qualitatively similar to *in vivo* experimental observations [2, 16, 9]). The LNP model fails to capture this fine temporal reproducibility. At the same time, the L-NLIF model is much more flexible and representationally powerful, as demonstrated in Fig. 2: by varying $V_r$ or $h$, for example, we can match a wide variety of dynamical behaviors (e.g. adaptation, bursting, bistability) known to exist in biological neurons.

## The Estimation Problem

Our problem now is to estimate the model parameters $\{\vec{k}, \sigma, g, V_r, h\}$ from a sufficiently rich, dynamic input sequence $\vec{x}(t)$ together with spike times $\{t_i\}$. A natural choice is the maximum likelihood estimator (MLE), which is easily proven to be consistent and statistically efficient here. To compute the MLE, we need to compute the likelihood and develop an algorithm for maximizing it.

The tractability of the likelihood function for this model arises directly from the linearity of the subthreshold dynamics of voltage $V(t)$ during an interspike interval. In the noiseless case [15], the voltage trace during an interspike interval $t \in [t_{i-1}, t_i]$ is given by the solution to equation (1) with $\sigma = 0$:

$$V_0(t) = V_r e^{-gt} + \int_{t_{i-1}}^{t} \left( \vec{k} \cdot \vec{x}(s) + \sum_{j=0}^{i-1} h(s - t_j) \right) e^{-g(t-s)} ds, \tag{2}$$

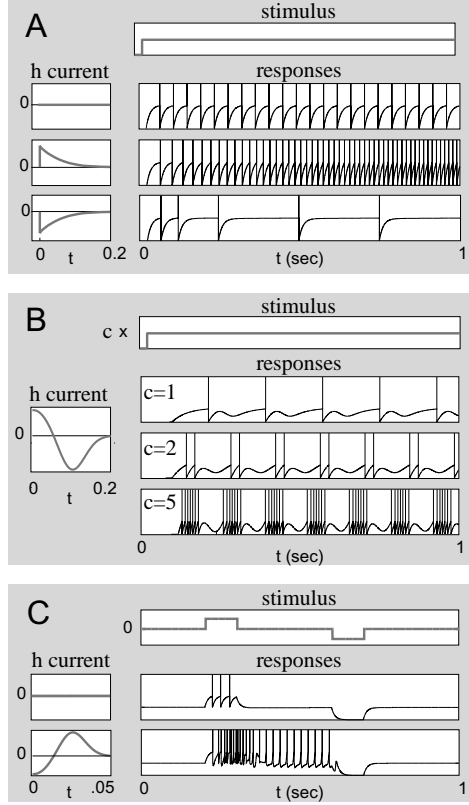

Figure 2: Illustration of diverse behaviors of L-NLIF model.

**A:** Firing rate adaptation. A positive DC current (top) was injected into three model cells differing only in their $h$ currents (shown on left: top, $h = 0$; middle, $h$ depolarizing; bottom, $h$ hyperpolarizing). Voltage traces of each cell's response (right, with spikes superimposed) exhibit rate facilitation for depolarizing $h$ (middle), and rate adaptation for hyperpolarizing $h$ (bottom).

**B:** Bursting. The response of a model cell with a biphasic $h$ current (left) is shown as a function of the three different levels of DC current. For small current levels (top), the cell responds rhythmically. For larger currents (middle and bottom), the cell responds with regular bursts of spikes.

**C:** Bistability. The stimulus (top) is a positive followed by a negative current pulse. Although a cell with no $h$ current (middle) responds transiently to the positive pulse, a cell with biphasic $h$ (bottom) exhibits a bistable response: the positive pulse puts it into a stable firing regime which persists until the arrival of a negative pulse.

which is simply a linear convolution of the input current with a negative exponential. It is easy to see that adding Gaussian noise to the voltage during each time step induces a Gaussian density over $V(t)$, since linear dynamics preserve Gaussianity [8]. This density is uniquely characterized by its first two moments; the mean is given by (2), and its covariance is $\sigma^2 E_g E_g^T$, where $E_g$ is the convolution operator corresponding to $e^{-gt}$. Note that this density is highly correlated for nearby points in time, since noise is integrated by the linear dynamics. Intuitively, smaller leak conductance $g$ leads to stronger correlation in $V(t)$ at nearby time points. We denote this Gaussian density $G(\vec{x}_i, \vec{k}, \sigma, g, V_r, h)$, where index $i$ indicates the $i$th spike and the corresponding stimulus chunk $\vec{x}_i$ (i.e. the stimuli that influence $V(t)$ during the $i$th interspike interval).

Now, on any interspike interval $t \in [t_{i-1}, t_i]$, the only information we have is that $V(t)$ is less than threshold for all times before $t_i$, and exceeds threshold during the time bin containing $t_i$. This translates to a set of linear constraints on $V(t)$, expressed in terms of the set

$$C_i = \bigcap_{t_{i-1} \leq t < t_i} \left\{ V(t) < 1 \right\} \cap \left\{ V(t_i) \geq 1 \right\}.$$

Therefore, the likelihood that the neuron first spikes at time $t_i$, given a spike at time $t_{i-1}$, is the probability of the event $V(t) \in C_i$, which is given by

$$L_{\vec{x}_i, t_i}(\vec{k}, \sigma, g, V_r, h) = \int_{C_i} G(\vec{x}_i, \vec{k}, \sigma, g, V_r, h),$$

the integral of the Gaussian density $G(\vec{x}_i, \vec{k}, \sigma, g, V_r, h)$ over the set $C_i$.

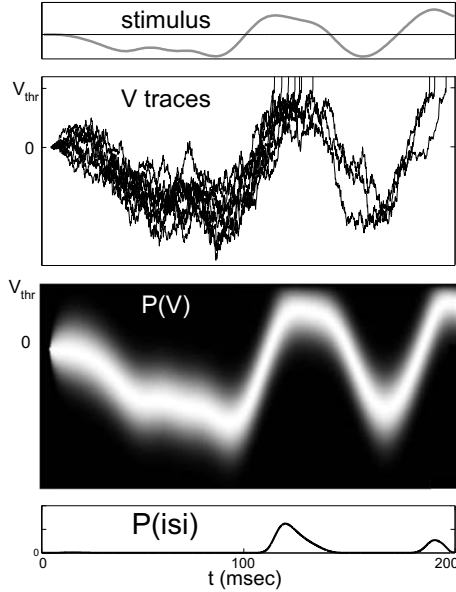

Figure 3: Behavior of the L-NLIF model during a single interspike interval, for a single (repeated) input current (top). **Top middle:** Ten simulated voltage traces $V(t)$, evaluated up to the first threshold crossing, conditional on a spike at time zero ($V_r = 0$). Note the strong correlation between neighboring time points, and the sparsening of the plot as traces are eliminated by spiking. **Bottom Middle:** Time evolution of $P(V)$. Each column represents the conditional distribution of $V$ at the corresponding time (i.e. for all traces that have not yet crossed threshold). **Bottom:** Probability density of the interspike interval (isi) corresponding to this particular input. Note that probability mass is concentrated at the points where input drives $V_0(t)$ close to threshold.

Spiking resets $V$ to $V_r$, meaning that the noise contribution to $V$ in different interspike intervals is independent. This "renewal" property, in turn, implies that the density over $V(t)$ for an entire experiment factorizes into a product of conditionally independent terms, where each of these terms is one of the Gaussian integrals derived above for a single interspike interval. The likelihood for the entire spike train is therefore the product of these terms over all observed spikes. Putting all the pieces together, then, the full likelihood is

$$L_{\{\vec{x}_i, t_i\}}(\vec{k}, \sigma, g, V_r, h) = \prod_i \int_{C_i} G(\vec{x}_i, \vec{k}, \sigma, g, V_r, h),$$

where the product, again, is over all observed spike times $\{t_i\}$ and corresponding stimulus chunks $\{\vec{x}_i\}$.

Now that we have an expression for the likelihood, we need to be able to maximize it. Our main result now states, basically, that we can use simple ascent algorithms to compute the MLE without getting stuck in local maxima.

**Theorem 1.** *The likelihood $L_{\{\vec{x}_i, t_i\}}(\vec{k}, \sigma, g, V_r, h)$ has no non-global extrema in the parameters $(\vec{k}, \sigma, g, V_r, h)$, for any data $\{\vec{x}_i, t_i\}$.*

The proof [14] is based on the log-concavity of $L_{\{\vec{x}_i, t_i\}}(\vec{k}, \sigma, g, V_r, h)$ under a certain parametrization of $(\vec{k}, \sigma, g, V_r, h)$. The classical approach for establishing the nonexistence of non-global maxima of a given function uses concavity, which corresponds roughly to the function having everywhere non-positive second derivatives. However, the basic idea can be extended with the use of any invertible function: if $f$ has no non-global extrema, neither will $g(f)$, for any strictly increasing real function $g$. The logarithm is a natural choice for $g$ in any probabilistic context in which independence plays a role, since sums are easier to work with than products. Moreover, concavity of a function $f$ is strictly stronger than logconcavity, so logconcavity can be a powerful tool even in situations for which concavity is useless (the Gaussian density is logconcave but not concave, for example). Our proof relies on a particular theorem [3] establishing the logconcavity of integrals of logconcave functions, and proceeds by making a correspondence between this type of integral and the

integrals that appear in the definition of the L-NLIF likelihood above.

We should also note that the proof extends without difficulty to some other noise processes which generate logconcave densities (where white noise has the standard Gaussian density); for example, the proof is nearly identical if $N_t$ is allowed to be colored or non-Gaussian noise, with possibly nonzero drift.

## Computational methods and numerical results

Theorem 1 tells us that we can ascend the likelihood surface without fear of getting stuck in local maxima. Now how do we actually compute the likelihood? This is a nontrivial problem: we need to be able to quickly compute (or at least approximate, in a rational way) integrals of multivariate Gaussian densities $G$ over simple but high-dimensional orthants $C_i$. We discuss two ways to compute these integrals; each has its own advantages.

The first technique can be termed "density evolution" [10, 13]. The method is based on the following well-known fact from the theory of stochastic differential equations [8]: given the data $(\vec{x}_i, t_{i-1})$, the probability density of the voltage process $V(t)$ up to the next spike $t_i$ satisfies the following partial differential (Fokker-Planck) equation:

$$\frac{\partial P(V,t)}{\partial t} = \frac{\sigma^2}{2} \frac{\partial^2 P}{\partial V^2} + g \frac{\partial[(V - V_{eq}(t))P]}{\partial V}, \tag{3}$$

under the boundary conditions

$$P(V, t_{i-1}) = \delta(V - V_r),$$

$$P(V_{th}, t) = 0;$$

where $V_{eq}(t)$ is the instantaneous equilibrium potential:

$$V_{eq}(t) = \frac{1}{g} \left( \vec{k} \cdot \vec{x}(t) + \sum_{j=0}^{i-1} h(t - t_j) \right).$$

Moreover, the conditional firing rate $f(t)$ satisfies

$$\int_{t_{i-1}}^{t} f(s)ds = 1 - \int P(V,t)dV.$$

Thus standard techniques for solving the drift-diffusion evolution equation (3) lead to a fast method for computing $f(t)$ (as illustrated in Fig. 2). Finally, the likelihood $L_{\vec{x}_i, t_i}(\vec{k}, \sigma, g, V_r, h)$ is simply $f(t_i)$.

While elegant and efficient, this density evolution technique turns out to be slightly more powerful than what we need for the MLE: recall that we do not need to compute the conditional rate function $f$ at all times $t$, but rather just at the set of spike times $\{t_i\}$, and thus we can turn to more specialized techniques for faster performance. We employ a rapid technique for computing the likelihood using an algorithm due to Genz [6], designed to compute exactly the kinds of multidimensional Gaussian probability integrals considered here. This algorithm works well when the orthants $C_i$ are defined by fewer than $\approx 10$ linear constraints on $V(t)$. The number of actual constraints on $V(t)$ during an interspike interval $(t_{i+1} - t_i)$ grows linearly in the length of the interval: thus, to use this algorithm in typical data situations, we adopt a strategy proposed in our work on the deterministic form of the model [15], in which we discard all but a small subset of the constraints. The key point is that, due to strong correlations in the noise and the fact that the constraints only figure significantly when the $V(t)$ is driven close to threshold, a small number of constraints often suffice to approximate the true likelihood to a high degree of precision.

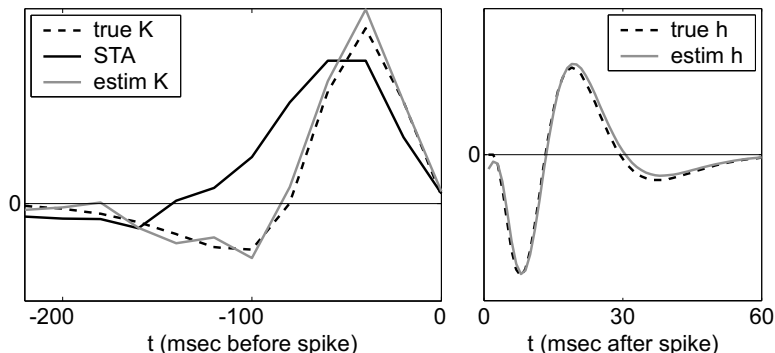

Figure 4: Demonstration of the estimator's performance on simulated data. Dashed lines show the true kernel $\vec{k}$ and aftercurrent $h$; $\vec{k}$ is a 12-sample function chosen to resemble the biphasic temporal impulse response of a macaque retinal ganglion cell, while $h$ is function specified in a five-dimensional vector space, whose shape induces a slight degree of bursti­ness in the model's spike responses. The L-NLIF model was stimulated with parameters $g = 0.05$ (corresponding to a membrane time constant of 20 time-samples), $\sigma_{noise} = 0.5$, and $V_r = 0$. The stimulus was 30,000 time samples of white Gaussian noise with a standard deviation of 0.5. With only 600 spikes of output, the estimator is able to retrieve an esti­mate of $\vec{k}$ (gray curve) which closely matches the true kernel. Note that the spike-triggered average (black curve), which is an unbiased estimator for the kernel of an LNP neuron [5], differs significantly from this true kernel (see also [15]).

The accuracy of this approach improves with the number of constraints considered, but performance is fastest with fewer constraints. Therefore, because ascending the likelihood function requires evaluating the likelihood at many different points, we can make this as­cent process much quicker by applying a version of the coarse-to-fine idea. Let $L_k$ denote the approximation to the likelihood given by allowing only $k$ constraints in the above al­gorithm. Then we know, by a proof identical to that of Theorem 1, that $L_k$ has no local maxima; in addition, by the above logic, $L_k \to L$ as $k$ grows. It takes little additional effort to prove that

$$\operatorname{argmax} L_k \to \operatorname{argmax} L;$$

thus, we can efficiently ascend the true likelihood surface by ascending the "coarse" ap­proximants $L_k$, then gradually "refining" our approximation by letting $k$ increase.

An application of this algorithm to simulated data is shown in Fig. 4. Further applications to both simulated and real data will be presented elsewhere.

**Discussion**

We have shown here that the L-NLIF model, which couples a linear filtering stage to a biophysically plausible and flexible model of neuronal spiking, can be efficiently estimated from extracellular physiological data using maximum likelihood. Moreover, this model lends itself directly to analysis via tools from the modern theory of point processes. For example, once we have obtained our estimate of the parameters $(\vec{k}, \sigma, g, V_r, h)$, how do we verify that the resulting model provides an adequate description of the data? This important "model validation" question has been the focus of some recent elegant research, under the rubric of "time rescaling" techniques [4]. While we lack the room here to review these methods in detail, we can note that they depend essentially on knowledge of the conditional firing rate function $f(t)$. Recall that we showed how to efficiently compute this function

in the last section and examined some of its qualitative properties in the L-NLIF context in Figs. 2 and 3.

We are currently in the process of applying the model to physiological data recorded both *in vivo* and *in vitro*, in order to assess whether it accurately accounts for the stimulus preferences and spiking statistics of real neurons. One long-term goal of this research is to elucidate the different roles of stimulus-driven and stimulus-independent activity on the spiking patterns of both single cells and multineuronal ensembles.

## Footnotes

* JWP and LP contributed equally to this work. We thank E.J. Chichilnisky for helpful discussions.

## References

[1] B. Aguera y Arcas and A. Fairhall. What causes a neuron to spike? *Neral Computation*, 15:1789–1807, 2003.

[2] M. Berry and M. Meister. Refractoriness and neural precision. *Journal of Neuroscience*, 18:2200–2211, 1998.

[3] V. Bogachev. *Gaussian Measures*. AMS, New York, 1998.

[4] E. Brown, R. Barbieri, V. Ventura, R. Kass, and L. Frank. The time-rescaling theorem and its application to neural spike train data analysis. *Neural Computation*, 14:325–346, 2002.

[5] E. Chichilnisky. A simple white noise analysis of neuronal light responses. *Network: Computation in Neural Systems*, 12:199–213, 2001.

[6] A. Genz. Numerical computation of multivariate normal probabilities. *Journal of Computational and Graphical Statistics*, 1:141–149, 1992.

[7] W. Gerstner and W. Kistler. *Spiking Neuron Models: Single Neurons, Populations, Plasticity*. Cambridge University Press, 2002.

[8] S. Karlin and H. Taylor. *A Second Course in Stochastic Processes*. Academic Press, New York, 1981.

[9] J. Keat, P. Reinagel, R. Reid, and M. Meister. Predicting every spike: a model for the responses of visual neurons. *Neuron*, 30:803–817, 2001.

[10] B. Knight, A. Omurtag, and L. Sirovich. The approach of a neuron population firing rate to a new equilibrium: an exact theoretical result. *Neural Computation*, 12:1045–1055, 2000.

[11] J. Levin and J. Miller. Broadband neural encoding in the cricket cercal sensory system enhanced by stochastic resonance. *Nature*, 380:165–168, 1996.

[12] L. Paninski. Convergence properties of some spike-triggered analysis techniques. *Network: Computation in Neural Systems*, 14:437–464, 2003.

[13] L. Paninski, B. Lau, and A. Reyes. Noise-driven adaptation: in vitro and mathematical analysis. *Neurocomputing*, 52:877–883, 2003.

[14] L. Paninski, J. Pillow, and E. Simoncelli. Maximum likelihood estimation of a stochastic integrate-and-fire neural encoding model. submitted manuscript (cns.nyu.edu/∼liam), 2004.

[15] J. Pillow and E. Simoncelli. Biases in white noise analysis due to non-poisson spike generation. *Neurocomputing*, 52:109–115, 2003.

[16] D. Reich, J. Victor, and B. Knight. The power ratio and the interval map: Spiking models and extracellular recordings. *The Journal of Neuroscience*, 18:10090–10104, 1998.

[17] M. Rudd and L. Brown. Noise adaptation in integrate-and-fire neurons. *Neural Computation*, 9:1047–1069, 1997.

[18] J. Victor. How the brain uses time to represent and process visual information. *Brain Research*, 886:33–46, 2000.

[19] Y. Yu and T. Lee. Dynamical mechanisms underlying contrast gain control in sing le neurons. *Physical Review E*, 68:011901, 2003.
